# Multiple Paired Forward-Inverse Models for Human Motor Learning and Control

**Masahiko Haruno**[*]
mharuno@hip.atr.co.jp

**Daniel M. Wolpert**[†]
wolpert@hera.ucl.ac.uk

**Mitsuo Kawato**[*°]
kawato@hip.atr.co.jp
[*]ATR Human Information Processing Research Laboratories
2-2 Hikaridai, Seika-cho, Soraku-gun, Kyoto 619-02, Japan.
[†]Sobell Department of Neurophysiology, Institute of Neurology,
Queen Square, London WC1N 3BG, United Kingdom.
[°]Dynamic Brain Project, ERATO, JST, Kyoto, Japan.

## Abstract

Humans demonstrate a remarkable ability to generate accurate and appropriate motor behavior under many different and often uncertain environmental conditions. This paper describes a new modular approach to human motor learning and control, based on multiple pairs of inverse (controller) and forward (predictor) models. This architecture simultaneously learns the multiple inverse models necessary for control as well as how to select the inverse models appropriate for a given environment. Simulations of object manipulation demonstrates the ability to learn multiple objects, appropriate generalization to novel objects and the inappropriate activation of motor programs based on visual cues, followed by on-line correction, seen in the "size-weight illusion".

## 1 Introduction

Given the multitude of contexts within which we must act, there are two qualitatively distinct strategies to motor control and learning. The first is to use a single controller which would need to be highly complex to allow for all possible scenarios. If this controller were unable to encapsulate all the contexts it would need to adapt every time the context of the movement changed before it could produce appropriate motor commands--this would produce transient and possibly large performance errors. Alternatively, a modular approach can be used in which multiple controllers co-exist, with each controller suitable for one or a small set of contexts. Such a modular strategy has been introduced in the "mixture of experts" architecture for supervised learning [6]. This architecture comprises a set of expert networks and a gating network which performs classification by combining each expert's output. These networks are trained simultaneously so that the gating network splits the input space into regions in which particular experts can specialize.

To apply such a modular strategy to motor control two problems must be solved. First

how are the set of inverse models (controllers) learned to cover the contexts which might be experienced the module learning problem. Second, given a set of inverse modules (controllers) how are the correct subset selected for the current context—the module selection problem. From human psychophysical data we know that such a selection process must be driven by two distinct processes; feedforward switching based on sensory signals such as the perceived size of an object, and switching based on feedback of the outcome of a movement. For example, on picking up a object which appears heavy, feedforward switching may activate controllers responsible for generating a large motor impulse. However, feedback processes, based on contact with the object, can indicate that it is in fact light thereby switching control to inverse models appropriate for a light object.

In the context of motor control and learning, Gomi and Kawato [4] combined the feedback-error-learning [7] approach and the mixture of experts architecture to learn multiple inverse models for different manipulated objects. They used both the visual shapes of the manipulated objects and intrinsic signals, such as somatosensory feedback and efference copy of the motor command, as the inputs to the gating network. Using this architecture it was quite difficult to acquire multiple inverse models. This difficulty arose because a single gating network needed to divide up, based solely on control error, the large input space into complex regions. Furthermore, Gomi and Kawato's model could not demonstrate feedforward controller selection prior to movement execution.

Here we describe a model of human motor control which addresses these problems and can solve the module learning and selection problems in a computationally coherent manner. The basic idea of the model is that the brain contains multiple pairs (modules) of forward (predictor) and inverse (controller) models (MPFIM) [10]. Within each module, the forward and inverse models are tightly coupled both during their acquisition and use, in which the forward models determine the contribution (responsibility) of each inverse model's output to the final motor command. This architecture can simultaneously learn the multiple inverse models necessary for control as well as how to select the inverse models appropriate for a given environment in both a feedforward and a feedback manner.

## 2 Multiple paired forward-inverse models

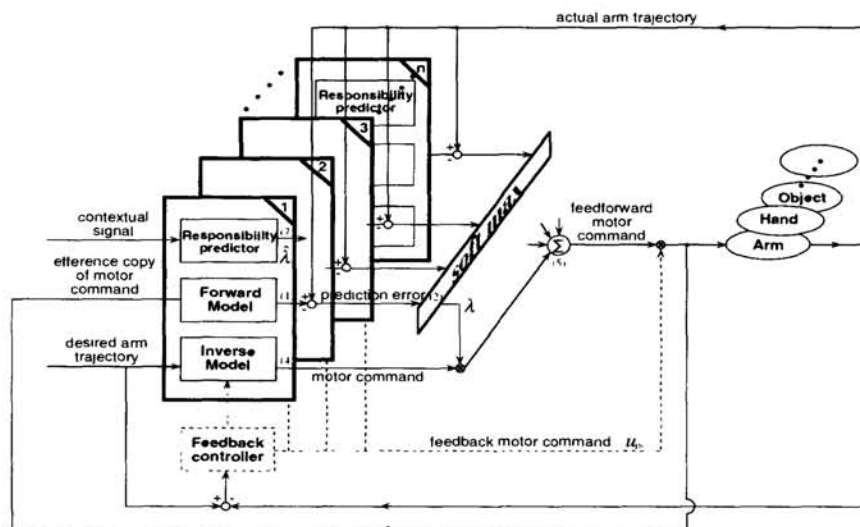

Figure 1: A schematic diagram showing how MPFIM architecture is used to control arm movement while manipulating different objects. Parenthesized numbers in the figure relate to the equations in the text.

## 2.1 Motor learning and feedback selection

Figure 1 illustrates how the MPFIM architecture can be used to learn and control arm movements when the hand manipulates different objects. Central to the multiple paired forward-inverse model is the notion of dividing up experience using predictive forward models. We consider $n$ undifferentiated forward models which each receive the current state, $x_t$, and motor command, $u_t$, as input. The output of the $i$th forward model is $\hat{x}_{t+1}^i$, the prediction of the next state at time $t$

$$\hat{x}_{t+1}^i = \phi(w_t^i, x_t, u_t) \tag{1}$$

where $w_t^i$ are the parameters of a function approximator $\phi$ (e.g. neural network weights) used to model the forward dynamics. These predicted next states are compared to the actual next state to provide the responsibility signal which represents the extent to which each forward model presently accounts for the behavior of the system. Based on the prediction errors of the forward models, the responsibility signal, $\lambda_t^i$, for the $i$-th forward-inverse model pair (module) is calculated by the soft-max function

$$\lambda_t^i = \frac{e^{-|x_t - \hat{x}_t^i|^2/2\sigma^2}}{\sum_{j=1}^n e^{-|x_t - \hat{x}_t^j|^2/2\sigma^2}} \tag{2}$$

where $x_t$ is the true state of the system and $\sigma$ is a scaling constant. The soft-max transforms the errors using the exponential function and then normalizes these values across the modules, so that the responsibilities lie between 0 and 1 and sum to 1 over the modules. Those forward models which capture the current behavior, and therefore produce small prediction errors, will have high responsibilities [1]. The responsibilities are then used to control the learning of the forward models in a competitive manner, with those models with high responsibilities receiving proportionally more of their error signal than modules with low responsibility. The competitive learning among forward models is similar in spirit to "annealed competition of experts" architecture [9].

$$\Delta w_t^i = \epsilon \lambda_t^i \frac{d\phi_t}{dw_t^i}(x_t - \hat{x}_t^i) = \epsilon \frac{d\hat{x}_t^i}{dw_t^i} \lambda_t^i (x_t - \hat{x}_t^i) \tag{3}$$

For each forward model there is a paired inverse model whose inputs are the desired next state $x_{t+1}^*$ and the current state $x_t$. The $i$th inverse model produces a motor command $u_t^i$ as output

$$u_t^i = \psi(\alpha_t^i, x_{t+1}^*, x_t) \tag{4}$$

where $\alpha_t^i$ are the parameters of some function approximator $\psi$.

The total motor command is the summation of the outputs from these inverse models using the responsibilities, $\lambda_t^i$, to weight the contributions.

$$u_t = \sum_{i=1}^n \lambda_t^i u_t^i = \sum_{i=1}^n \lambda_t^i \psi(\alpha_t^i, x_{t+1}^*, x_t) \tag{5}$$

Once again, the responsibilities are used to weight the learning of each inverse model. This ensures that inverse models learns only when their paired forward models make accurate predictions. Although for supervised learning the desired control command $u_t^*$ is needed (but is generally not available), we can approximate $(u_t^* - u_t^i)$ with the feedback motor command signal $u_{fb}$ [7].

$$\Delta\alpha_t^i = \epsilon\lambda_t^i\frac{d\psi_i}{d\alpha_t^i}(u_t^* - u_t^i) = \epsilon\frac{du_t^i}{d\alpha_t^i}\lambda_t^i(u_t^* - u_t^i) \simeq \epsilon\frac{du_t^i}{d\alpha_t^i}\lambda_t^i u_{fb} \qquad (6)$$

In summary, the responsibility signals are used in three ways—first to gate the learning of the forward models (Equation 3), second to gate the learning of the inverse models (Equation 6), and third to gate the contribution of the inverse models to the final motor command (Equation 5).

## 2.2 Multiple responsibility predictors: Feedforward selection

While the system described so far can learn multiple controllers and switch between them based on prediction errors, it cannot provide switching before a motor command has been generated and the consequences of this action evaluated. To allow the system to switch controllers based on contextual information, we introduce a new component, the responsibility predictor (RP). The input to this module, $y_t$, contains contextual sensory information (Figure 1) and each RP produces a prediction of its own module's responsibility

$$\hat{\lambda}_t^i = \eta(\gamma_t^i, y_t). \qquad (7)$$

These estimated responsibilities can then be compared to the actual responsibilities $\lambda_t^i$ generated from the responsibility estimator. These error signals are used to update the weights of the RP by supervised learning.

Finally a mechanism is required to combine the responsibility estimates derived from the feedforward RP and from the forward models' prediction errors derived from feedback. We determine the final value of responsibility by using Bayes rule; multiplying the transformed feedback errors $e^{-|x_t - \hat{x}_t^i|^2/\sigma^2}$ by the feedforward responsibility $\hat{\lambda}_t^i$ and then normalizing across the modules within the responsibility estimator: $\hat{\lambda}_t^i e^{-|x_t - \hat{x}_t^i|^2/2\sigma^2} / \sum_{j=1}^n \hat{\lambda}_t^j e^{-|x_t - \hat{x}_t^j|^2/2\sigma^2}$

The estimates of the responsibilities produced by the RP can be considered as *prior* probabilities because they are computed before the movement execution based only on extrinsic signals and do not rely on knowing the consequences of the action. Once an action takes place, the forward models' errors can be calculated and this can be thought of as the *likelihood* after the movement execution based on knowledge of the result of the movement. The final responsibility which is the product of the prior and likelihood, normalized across the modules, represents the *posterior* probability. Adaptation of the RP ensures that the prior probability becomes closer to the posterior probability.

# 3 Simulation of arm tracking while manipulating objects

## 3.1 Learning and control of different objects

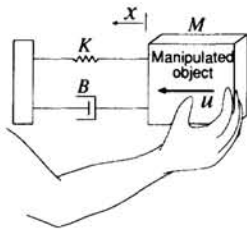

|   | $M$ (Kg) | $B$ (N m$^{-1}$ s) | $K$ (N m$^{-1}$) |
|---|---|---|---|
| $\alpha$ | 1.0 | 2.0 | 8.0 |
| $\beta$ | 5.0 | 7.0 | 4.0 |
| $\gamma$ | 8.0 | 3.0 | 1.0 |
| $\delta$ | 2.0 | 10.0 | 1.0 |

Figure 2: Schematic illustration of the simulation experiment in which the arm makes reaching movements while grasping different objects with mass $M$, damping $B$ and spring $K$. The object properties are shown in the Table.

To examine motor learning and control we simulated a task in which the hand had to track a given trajectory (30 s shown in Fig. 3 (b)), while holding different objects (Figure 2). The manipulated object was periodically switched every 5 s between three different objects $\alpha$, $\beta$ and $\gamma$ in this order. The physical characteristics of these objects are shown in Figure 2. The task was exactly the same as that of Gomi and Kawato [4], and simulates recent grip force-load force coupling experiments by Flanagan and Wing [2].

In the first simulation, three forward-inverse model pairs (modules) were used: the same number of modules as the number of objects. We assumed the existence of a perfect inverse dynamic model of the arm for the control of reaching movements. In each module, both forward ($\phi$ in (1)) and inverse ($\psi$ in (4)) models were implemented as a linear neural network[2]. The use of linear networks allowed $M$, $B$ and $K$ to be estimated from the forward and inverse model weights. Let $M_j^F, B_j^F, K_j^F$ be the estimates from the $j$th forward model and $M_j^I, B_j^I, K_j^I$ be the estimates from the $j$th inverse model.

Figure 3(a) shows the evolution of the forward model estimates of $M_j^F, B_j^F, K_j^F$ for the three modules during learning. During learning the desired trajectory (Fig. 3(b)) was repeated 200 times. The three modules started from randomly selected initial conditions (open arrows) and converged to very good approximations of the three objects (filled arrows) as shown in Table 1. Each of the three modules converged to $\alpha$, $\beta$ and $\gamma$ objects, respectively. It is interesting to note that all the estimates of the forward models are superior to those of inverse models. This is because the inverse model learning depends on how modules are switched by the forward models.

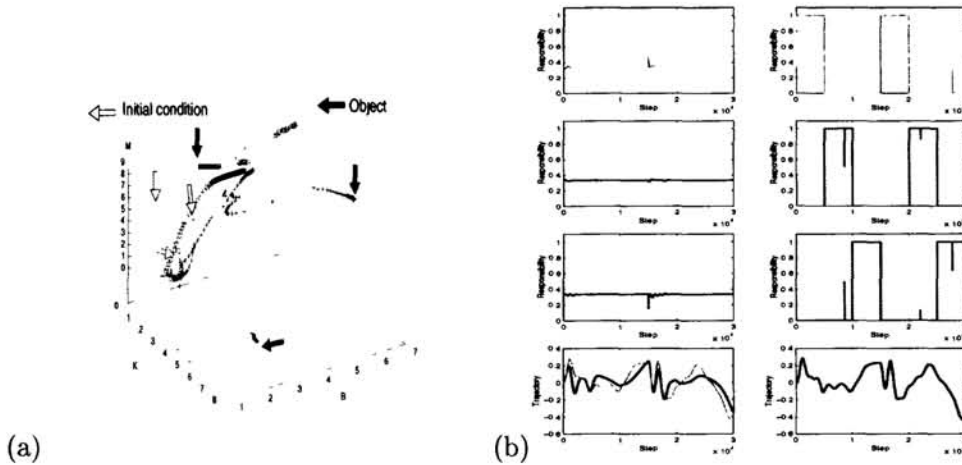

(a)                                                           (b)

Figure 3: (a) Learning acquisition of three pairs of forward and inverse models corresponding to three objects. (b) Responsibility signals from the three modules (top 3) and tracking performance (bottom) at the beginning (left) and at the end (right) of learning.

| Module | $M_j^F$ | $B_j^F$ | $K_j^F$ | $M_j^I$ | $B_j^I$ | $K_j^I$ |
|--------|---------|---------|---------|---------|---------|---------|
| 1 | 1.0020 | 2.0080 | 8.0000 | 1.0711 | 2.0080 | 8.0000 |
| 2 | 5.0071 | 7.0040 | 4.0000 | 5.0102 | 6.9554 | 4.0089 |
| 3 | 8.0029 | 3.0010 | 0.9999 | 7.8675 | 3.0467 | 0.9527 |

Table 1: Learned object characteristics

Figure 3(b) shows the performance of the model at the beginning (left) and end (right) of learning. The top 3 panels show the responsibility signals of $\alpha$, $\beta$ and $\gamma$ modules in

this order, and the bottom panel shows the hand's actual and desired trajectories. At the start of learning, the three modules were equally poor and thus generated almost equal responsibilities (1/3) and were involved in control almost equally. As a result, the overall control performance was poor with large trajectory errors. However, at the end of learning, the three modules switched almost perfectly (only three noisy spikes were observed in the top 3 panels on the right), and no trajectory error was visible at this resolution in the bottom panel. If we compare these results with Figure 7 of Gomi and Kawato [4] for the same task, the superiority of the MPFIM compared to the gating-expert architecture is apparent. Note that the number of free parameters (synaptic weights) is smaller in the current architecture than the other. The difference in performance comes from two features of the basic architecture. First, in the gating architecture a single gating network tries to divide the space while many forward models splits the space in MPFIM. Second, in the gating architecture only a single control error is used to divide the space, but multiple prediction errors are simultaneously utilized in MPFIM.

## 3.2   Generalization to a novel object

A natural question regarding MPFIM architecture is how many modules need to be used. In other words, what happens if the number of objects exceeds the number of modules or an already trained MPFIM is presented with an unfamiliar object. To examine this, the MPFIM trained from 4 objects $\alpha,\beta,\gamma$ and $\delta$ was presented with a novel object $\eta$ (its $(M, B, K)$ is $(2.02, 3.23, 4.47)$). Because the object dynamics can be represented in a 3-dimensional parameter space and the 4 modules already acquired define 4 vertices of a tetrahedron within the 3-D space, arbitrary object dynamics contained within the tetrahedron can be decomposed into a weighted average of the existing 4 forward modules (internal division point of the 4 vertices). The theoretically calculated weights of $\eta$ were $(0.15, 0.20, 0.35, 0.30)$. Interestingly, each module's responsibility signal averaged over trajectory was $(0.14, 0.24, 0.37, 0.26)$. Although the responsibility was computed in the space of accelerations prediction by soft-max and had no direct relation to the space of $(M, B, K)$, the two vectors had very similar values. This demonstrates the flexibility of MPFIM architecture which originates from its probabilistic soft-switching mechanism. This is in sharp contrast to the hard switching of Narendra [8] for which only one controller can be selected at a time.

## 3.3   Feedforward selection and the size-weight illusion

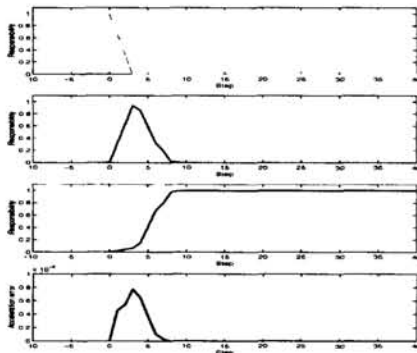

Figure 4: Responsibility predictions based on contextual information of 2-D object shapes (top 3 traces) and corresponding acceleration error of control induced by the illusion (bottom trace)

In this section, we simulated prior selection of inverse models by responsibility predictors based on contextual information, and reproduce the size-weight illusion. Each object was associated with a 2-D shape represented as a 3×3 binary matrix, which was randomly placed at one of four possible locations on a 4×4 retinal matrix (see Gomi

and Kawato for more details). The retinal matrix was used as the contextual input to the RP (3-layer sigmoidal feedforward network). During the course of learning, the combination of manipulated objects and visual cues were fixed as A-$\alpha$, B-$\beta$ and C-$\gamma$. After 200 iterations of the trajectory, the combination A-$\gamma$ was presented for the first. Figure 4 plots the responsibility signals of the three modules (top 3 traces) and corresponding acceleration error of the control induced by the illusion (bottom trace). The result replicates the size-weight illusion [1, 5] seen in the erroneous responsibility prediction of the $\alpha$ responsibility predictor based on the contextual signal A and its correction by the responsibility signal calculated by the forward models. Until the onset of movement (time 0), $A$ was always associated with light $\alpha$, and $C$ was always associated with heavy $\gamma$. Prior to movement when $A$ was associated with $\gamma$, the $\alpha$ module was switched on by the visual contextual information, but soon after the movement was initiated, the responsibility signal from the forward model's prediction dominated, and the $\gamma$ module was properly selected. Furthermore, after a while, the responsibility predictor of the modules were re-learned to capture this new association between the objects visual shape and its dynamics.

In conclusion, the MPFIM model of human motor learning and control, like the human motor system, can learn multiple tasks, shows generalization to new tasks and an ability to switch between tasks appropriately.

## Acknowledgments

We thank Zoubin Ghahramani for helpful discussions on the Bayesian formulation of this model. Partially supported by Special Coordination Funds for promoting Science and Technology at the Science and Technology Agency of Japanese govenmnent, and by HFSP grant.

## Footnotes

[1]Because selecting modules can be regarded as a hidden state estimation problem, an alternative way to determine appropriate forward models is to use the EM algorithm [3].

[2]Any kind of architecture can be adopted instead of linear networks

## References

[1] E. Brenner, B. Jeroen, and J. Smeets. Size illusion influences how we lift but not how we grasp an object. *Exp Brain Res*, 111:473–476, 1996.

[2] J.R. Flanagan and A. Wing. The role of internal models in motion planning and control: Evidence from grip force adjustments during movements of hand-held loads. *J Neurosci*, 17(4):1519–1528, 1997.

[3] A.M. Fraser and A. Dimitriadis. Forecasting probability densities by using hidden Markov models with mixed states. In A.S. Wiegand and N.A. Gershenfeld, editors, *Time series prediction: Forecasting the future and understanding the past*, pages 265–282. Addison-Wesley, 1993.

[4] H. Gomi and M. Kawato. Recognition of manipulated objects by motor learning with modular architecture networks. *Neural Networks*, 6:485–497, 1993.

[5] A. Gordon, H. Forssberg, R. Johansson, and G. Westling. Visual size cues in the programming of manipulative forces during precision grip. *Exp Brain Res*, 83:477–482, 1991.

[6] R. Jacobs, M. Jordan, S. Nowlan, and G. Hinton. Adaptive mixture of local experts. *Neural Computation*, 3:79–87, 1991.

[7] M. Kawato. Feedback-error-learning neural network for supervised learning. In R. Eckmiller, editor, *Advanced neural computers*, pages 365–372. North-Holland, 1990.

[8] K. Narendra and J. Balakrishnan. Adaptive control using multiple models. *IEEE Transaction on Automatic Control*, 42(2):171–187, 1997.

[9] K. Pawelzik, J. Kohlmorgen, and K. Müller. Annealed competition of experts for a segmentation and classification of switching dynamics. *Neural Computation*, 8:340–356, 1996.

[10] D.M. Wolpert and M. Kawato. Multiple paired forward and inverse models for motor control. *Neural Networks*, 11:1317–1329, 1998.